# Reinforcement Learning with Hierarchies of Machines *

**Ronald Parr and Stuart Russell**
Computer Science Division, UC Berkeley, CA 94720
{parr,russell}@cs.berkeley.edu

## Abstract

We present a new approach to reinforcement learning in which the policies considered by the learning process are constrained by hierarchies of partially specified machines. This allows for the use of prior knowledge to reduce the search space and provides a framework in which knowledge can be transferred across problems and in which component solutions can be recombined to solve larger and more complicated problems. Our approach can be seen as providing a link between reinforcement learning and "behavior-based" or "teleo-reactive" approaches to control. We present provably convergent algorithms for problem-solving and learning with hierarchical machines and demonstrate their effectiveness on a problem with several thousand states.

## 1 Introduction

Optimal decision making in virtually all spheres of human activity is rendered intractable by the complexity of the task environment. Generally speaking, the only way around intractability has been to provide a hierarchical organization for complex activities. Although it can yield suboptimal policies, top-down hierarchical control often reduces the complexity of decision making from exponential to linear in the size of the problem. For example, hierarchical task network (HTN) planners can generate solutions containing tens of thousands of steps [5], whereas "flat" planners can manage only tens of steps.

HTN planners are successful because they use a plan library that describes the decomposition of high-level activities into lower-level activities. This paper describes an approach to learning and decision making in *uncertain* environments (Markov decision processes) that uses a roughly analogous form of prior knowledge. We use *hierarchical abstract machines* (HAMs), which impose constraints on the policies considered by our learning algorithms. HAMs consist of nondeterministic finite state machines whose transitions may invoke lower-level machines. Nondeterminism is represented by *choice states* where the optimal action is yet to be decided or learned. The language allows a variety of prior constraints to be expressed, ranging from no constraint all the way to a fully specified solution. One

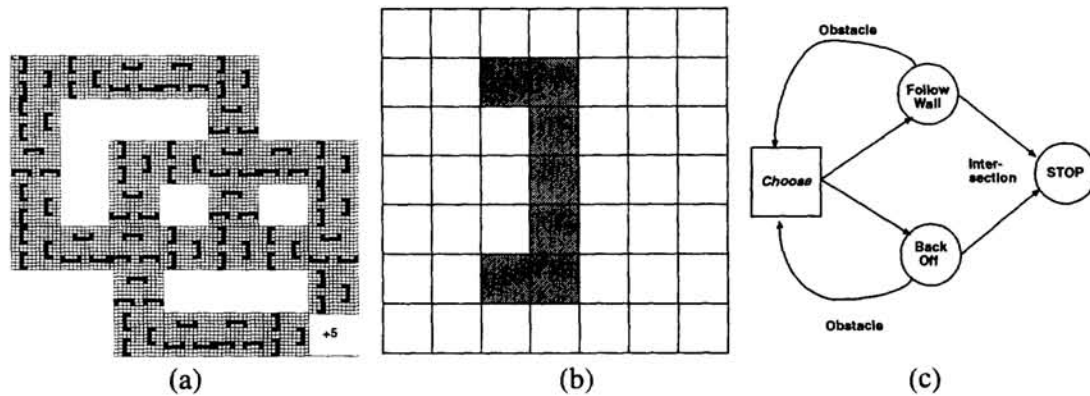

|       |       |       |
| ----- | ----- | ----- |
| (a)   | (b)   | (c)   |

Figure 1: (a) An MDP with $\approx 3600$ states. The initial state is in the top left. (b) Close-up showing a typical obstacle. (c) Nondeterministic finite-state controller for negotiating obstacles.

useful intermediate point is the specification of just the general organization of behavior into a layered hierarchy, leaving it up to the learning algorithm to discover exactly which lower-level activities should be invoked by higher levels at each point.

The paper begins with a brief review of Markov decision processes (MDPs) and a description of hierarchical abstract machines. We then present, in abbreviated form, the following results: **1)** Given any HAM and any MDP, there exists a new MDP such that the optimal policy in the new MDP is optimal in the original MDP among those policies that satisfy the constraints specified by the HAM. This means that even with complex machine specifications we can still apply standard decision-making and learning methods. **2)** An algorithm exists that determines this optimal policy, given an MDP and a HAM. **3)** On an illustrative problem with 3600 states, this algorithm yields dramatic performance improvements over standard algorithms applied to the original MDP. **4)** A reinforcement learning algorithm exists that converges to the optimal policy, subject to the HAM constraints, with no need to construct explicitly a new MDP. **5)** On the sample problem, this algorithm learns dramatically faster than standard RL algorithms. We conclude with a discussion of related approaches and ongoing work.

## 2   Markov Decision Processes

We assume the reader is familiar with the basic concepts of MDPs. To review, an MDP is a 4-tuple, $(\mathcal{S}, \mathcal{A}, T, R)$ where $\mathcal{S}$ is a set of *states*, $\mathcal{A}$ is a set of *actions*, $T$ is a *transition model* mapping $\mathcal{S} \times \mathcal{A} \times \mathcal{S}$ into probabilities in $[0, 1]$, and $R$ is a *reward function* mapping $\mathcal{S} \times \mathcal{A} \times \mathcal{S}$ into real-valued rewards. Algorithms for solving MDPs can return a *policy* $\pi$ that maps from $\mathcal{S}$ to $\mathcal{A}$, a real-valued *value function* $V$ on states, or a real-valued $Q$-function on state–action pairs. In this paper, we focus on infinite-horizon MDPs with a discount factor $\beta$. The aim is to find an optimal policy $\pi^*$ (or, equivalently, $V^*$ or $Q^*$) that maximizes the expected discounted total reward of the agent.

Throughout the paper, we will use as an example the MDP shown in Figure 1(a). Here $\mathcal{A}$ contains four primitive actions (up, down, left, right). The transition model, $T$, specifies that each action succeeds 80% of time, while 20% of the time the agent moves in an unintended perpendicular direction. The agent begins in a start state in the upper left corner. A reward of 5.0 is given for reaching the goal state and the discount factor $\beta$ is 0.999.

## 3   Hierarchical abstract machines

A HAM is a program which, when executed by an agent in an environment, constrains the actions that the agent can take in each state. For example, a very simple machine might dictate, "repeatedly choose right or down," which would eliminate from consideration all policies that go up or left. HAMs extend this simple idea of constraining policies by providing a hierarchical means of expressing constraints at varying levels of detail and

specificity. Machines for HAMs are defined by a set of states, a transition function, and a start function that determines the initial state of the machine. Machine states are of four types: **Action** states execute an action in the environment. **Call** states execute another machine as a subroutine. **Choice** states nondeterministically select a next machine state. **Stop** states halt execution of the machine and return control to the previous call state.

The transition function determines the next machine state after an action or call state as a stochastic function of the current machine state and some features of the resulting environment state. Machines will typically use a partial description of the environment to determine the next state. Although machines can function in partially observable domains, for the purposes of this paper we make the standard assumption that the agent has access to a complete description as well.

A HAM is defined by an initial machine in which execution begins and the closure of all machines reachable from the initial machine. Figure 1(c) shows a simplified version of one element of the HAM we used for the MDP in Figure 1. This element is used for traversing a hallway while negotiating obstacles of the kind shown in Figure 1(b). It runs until the end of the hallway or an intersection is reached. When it encounters an obstacle, a *choice point* is created to choose between two possible next machine states. One calls the backoff machine to back away from the obstacle and then move forward until the next one. The other calls the follow-wall machine to try to get around the obstacle. The follow-wall machine is very simple and will be tricked by obstacles that are concave in the direction of intended movement; the backoff machine, on the other hand, can move around any obstacle in this world but could waste time backing away from some obstacles unnecessarily and should be used sparingly.

Our complete "navigation HAM" involves a three-level hierarchy, somewhat reminiscent of a Brooks-style architecture but with hard-wired decisions replaced by choice states. The top level of the hierarchy is basically just a choice state for choosing a hallway navigation direction from the four coordinate directions. This machine has control initially and regains control at intersections or corners. The second level of the hierarchy contains four machines for moving along hallways, one for each direction. Each machine at this level has a choice state with four basic strategies for handling obstacles. Two back away from obstacles and two attempt to follow walls to get around obstacles. The third level of the hierarchy implements these strategies using the primitive actions.

The transition function for this HAM assumes that an agent executing the HAM has access to a short-range, low-directed sonar that detects obstacles in any of the four axis-parallel adjacent squares and a long-range, high-directed sonar that detects larger objects such as the intersections and the ends of hallways. The HAM uses these partial state descriptions to identify feasible choices. For example, the machine to traverse a hallway northwards would not be called from the start state because the high-directed sonar would detect a wall to the north.

Our navigation HAM represents an abstract plan to move about the environment by repeatedly selecting a direction and pursuing this direction until an intersection is reached. Each machine for navigating in the chosen direction represents an abstract plan for moving in a particular direction while avoiding obstacles. The next section defines how a HAM interacts with a specific MDP and how to find an optimal policy that respects the HAM constraints.

## 4 Defining and solving the HAM-induced MDP

A policy for a model, $M$, that is *HAM-consistent* with HAM $H$ is a scheme for making choices whenever an agent executing $H$ in $M$, enters a choice state. To find the optimal HAM-consistent policy we *apply* $H$ to $M$ to yield an *induced* MDP, $H \circ M$. A somewhat simplified description of the construction of $H \circ M$ is as follows: **1)** The set of states in $H \circ M$ is the cross-product of the states of $H$ with the states of $M$. **2)** For each state in $H \circ M$ where the machine component is an action state, the model and machine transition

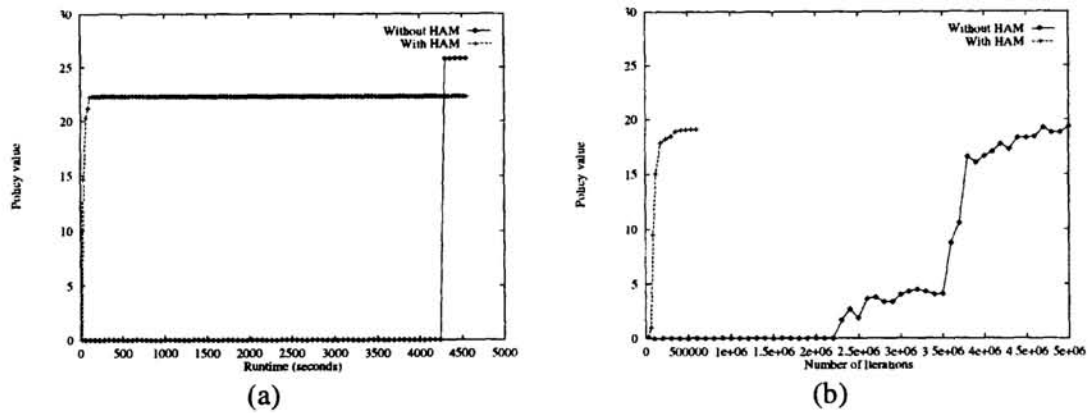

Figure 2: Experimental results showing policy value (at the initial state) as a function of runtime on the domain shown in Figure 1. (a) Policy iteration with and without the HAM. (b) Q-learning with and without the HAM (averaged over 10 runs).

functions are combined. **3)** For each state where the machine component is a choice state, actions that change only the machine component of the state are introduced. **4)** The reward is taken from $M$ for primitive actions, otherwise it is zero. With this construction, we have the following (proof omitted):

**Lemma 1** For any Markov decision process $M$ and any[1] HAM $H$, the induced process $H \circ M$ is a Markov decision process.

**Lemma 2** If $\pi$ is an optimal policy for $H \circ M$, then the primitive actions specified by $\pi$ constitute the optimal policy for $M$ that is HAM-consistent with $H$.

Of course, $H \circ M$ may be quite large. Fortunately, there are two things that will make the problem much easier in most cases. The first is that not all pairs of HAM states and environment states will be possible, i.e., reachable from an initial state. The second is that the actual complexity of the induced MDP is determined by the number of choice points, i.e., states of $H \circ M$ in which the HAM component is a choice state. This leads to the following:

**Theorem 1** For any MDP, $M$, and HAM, $H$, let $C$ be the set of choice points in $H \circ M$. There exists a decision process, $reduce(H \circ M)$, with states $C$ such that the optimal policy for $reduce(H \circ M)$ corresponds to the optimal policy for $M$ that is HAM-consistent with $H$.

**Proof sketch** We begin by applying Lemma 1 and then observing that in states of $H \circ M$ where the HAM component is not a choice state, only one action is permitted. These states can be removed to produce an equivalent Semi-Markov decision process (SMDP). (SMDPs are a generalization of Markov decision processes that permit different discount rates for different transitions.) The optimal policy for this SMDP will be the same as the optimal policy for $H \circ M$ and by Lemma 2, this will be the optimal policy for $M$ that is HAM-consistent with H. $\square$

This theorem formally establishes the mechanism by which the constraints embodied in a HAM can be used to simplify an MDP. As an example of the power of this theorem, and to demonstrate that this transformation can be done efficiently, we applied our navigation HAM to the problem described in the previous section. Figure 2(a) shows the results of applying policy iteration to the original model and to the transformed model. Even when we add in the cost of transformation (which, with our rather underoptimized code, takes

866 seconds), the HAM method produces a good policy in less than a quarter of the time required to find the optimal policy in the original model. The actual solution time is 185 seconds versus 4544 seconds.

An important property of the HAM approach is that model transformation produces an MDP that is an accurate model of the application of the HAM to the original MDP. Unlike typical approximation methods for MDPs, the HAM method can give strict performance guarantees. The solution to the transformed model $reduce(H \circ M)$ is the optimal solution from within a well-defined class of policies and the value assigned to this solution is the true expected value of applying the concrete HAM policy to the original MDP.

## 5 Reinforcement learning with HAMs

HAMs can be of even greater advantage in a reinforcement learning context, where the effort required to obtain a solution typically scales very badly with the size of the problem. HAM contraints can focus exploration of the state space, reducing the "blind search" phase that reinforcement learning agents must endure while learning about a new environment. Learning will also be faster for the same reason policy iteration is faster in the HAM-induced model; the agent is effectively operating in a reduced state space.

We now introduce a variation of Q-learning called HAMQ-learning that learns directly in the reduced state space *without performing the model transformation* described in the previous section. This is significant because the the environment model is not usually known *a priori* in reinforcement learning contexts.

A HAMQ-learning agent keeps track of the following quantities: $t$, the current environment state; $n$, the current machine state; $s_c$ and $m_c$, the environment state and machine state at the previous choice point; $a$, the choice made at the previous choice point; and $r_c$ and $\beta_c$, the total accumulated reward and discount since the previous choice point. It also maintains an extended Q-table, $Q([s, m], a)$, which is indexed by an environment-state/machine-state pair and by an action taken at a choice point.

For every environment transition from state $s$ to state $t$ with observed reward $r$ and discount $\beta$, the HAMQ-learning agent updates: $r_c \leftarrow r_c + \beta_c r$ and $\beta_c \leftarrow \beta\beta_c$. For each transition to a choice point, the agent does

$$Q([s_c, m_c], a) \leftarrow Q([s_c, m_c], a) + \alpha[r_c + \beta_c V([t, n]) - Q([s_c, m_c], a)],$$

and then $r_c \leftarrow 0, \beta_c \leftarrow 1$.

**Theorem 2** For any finite-state MDP, $M$, and any HAM, $H$, HAMQ-learning will converge to the optimal choice for every choice point in $reduce(H \circ M)$ with probability 1.

**Proof sketch** We note that the expected reinforcement signal in HAMQ-learning is the same as the expected reinforcement signal that would be received if the agent were acting directly in the transformed model of Theorem 1 above. Thus, Theorem 1 of [11] can be applied to prove the convergence of the HAMQ-learning agent, provided that we enforce suitable constraints on the exploration strategy and the update parameter decay rate.□

We ran some experiments to measure the performance of HAMQ-learning on our sample problem. Exploration was achieved by selecting actions according to the Boltzman distribution with a temperature parameter for each state. We also used an inverse decay for the update parameter $\alpha$. Figure 2(b) compares the learning curves for Q-learning and HAMQ-learning. HAMQ-learning appears to learn much faster: Q-learning required 9,000,000 iterations to reach the level achieved by HAMQ-learning after 270,000 iterations. Even after 20,000,000 iterations, Q-learning did not do as well as HAMQ-learning.[2]

# 6   Related work

*State aggregation* (see, e.g., [18] and [7]) clusters "similar" states together and assigns them the same value, effectively reducing the state space. This is orthogonal to our approach and could be combined with HAMs. However, aggregation should be used with caution as it treats distinct states as a single state and can violate the Markov property leading to the loss of performance guarantees and oscillation or divergence in reinforcement learning. Moreover, state aggregation may be hard to apply effectively in many cases.

Dean and Lin [8] and Bertsekas and Tsitsiklis [2], showed that some MDPs are loosely coupled and hence amenable to divide-and-conquer algorithms. A machine-like language was used in [13] to partition an MDP into decoupled subproblems. In problems that are amenable to decoupling, this could approaches could be used in combinated with HAMs.

Dayan and Hinton [6] have proposed *feudal* RL which specifies an explicit subgoal structure, with fixed values for each subgoal achieved, in order to achieve a hierarchical decomposition of the state space. Dietterich extends and generalizes this approach in [9]. Singh has investigated a number of approaches to subgoal based decomposition in reinforcement learning (e.g. [17] and [16]). Subgoals seem natural in some domains, but they may require a significant amount of outside knowledge about the domain and establishing the relationship between the value of subgoals with respect to the overall problem can be difficult.

Bradtke and Duff [3] proposed an RL algorithm for SMDPs. Sutton [19] proposes *temporal abstractions*, which concatenate sequences of state transitions together to permit reasoning about temporally extended events, and which can thereby form a behavioral hierarchy as in [14] and [15]. Lin's somewhat informal scheme [12] also allows agents to treat entire policies as single actions. These approaches can be emcompassed within our framework by encoding the events or behaviors as machines.

The design of hierarchically organized, "layered" controllers was popularized by Brooks [4]. His designs use a somewhat different means of passing control, but our analysis and theorems apply equally well to his machine description language. The "teleo-reactive" agent designs of Benson and Nilsson [1] are even closer to our HAM language. Both of these approaches assume that the agent is completely specified, albeit self-modifiable. The idea of partial behavior descriptions can be traced at least to Hsu's *partial programs* [10], which were used with a deterministic logical planner.

# 7   Conclusions and future work

We have presented HAMs as a principled means of constraining the set of policies that are considered for a Markov decision process and we have demonstrated the efficacy of this approach in a simple example for both policy iteration and reinforcement learning. Our results show very significant speedup for decision-making and learning—but of course, this reflects the provision of knowledge in the form of the HAM. The HAM language provides a very general method of transferring knowledge to an agent and we only have scratched the surface of what can be done with this approach.

We believe that if desired, subgoal information can be incorporated into the HAM structure, unifying subgoal-based approaches with the HAM approach. Moreover, the HAM structure provides a natural decomposition of the HAM-induced model, making it amenable to the divide-and-conquer approaches of [8] and [2].

There are opportunities for generalization across all levels of the HAM paradigm. Value function approximation can be used for the HAM induced model and inductive learning methods can be used to produce HAMs or to generalize their effects upon different regions of the state space. Gradient-following methods also can be used to adjust the transition probabilities of a stochastic HAM.

HAMs also lend themselves naturally to partially observable domains. They can be applied directly when the choice points induced by the HAM are states where no confusion about

the true state of the environment is possible. The application of HAMs to more general partially observable domains is more complicated and is a topic of ongoing research. We also believe that the HAM approach can be extended to cover the average-reward optimality criterion.

We expect that successful pursuit of these lines of research will provide a formal basis for understanding and unifying several seemingly disparate approaches to control, including behavior-based methods. It should also enable the use of the MDP framework in real-world applications of much greater complexity than hitherto attacked, much as HTN planning has extended the reach of classical planning methods.

## Footnotes

*This research was supported in part by ARO under the MURI program "Integrated Approach to Intelligent Systems," grant number DAAH04-96-1-0341.

[1]To preserve the Markov property, we require that if a machine has more than one possible caller in the hierarchy, that each appearance is treated as a distinct machine. This is equivalent to requiring that the call graph for the HAM is a tree. It follows from this that circular calling sequences are also forbidden.

[2]Speedup techniques such as eligibility traces could be applied to get better Q-learning results; such methods apply equally well to HAMQ-learning.

## References

[1] S. Benson and N. Nilsson. Reacting, planning and learning in an autonomous agent. In K. Furukawa, D. Michie, and S. Muggleton, editors, *Machine Intelligence 14*. Oxford University Press, Oxford, 1995.

[2] D. C. Bertsekas and J. N. Tsitsiklis. *Parallel and Distributed Computation: Numerical Methods*. Prentice-Hall, Englewood Cliffs, New Jersey, 1989.

[3] S. J. Bradtke and M. O. Duff. Reinforcement learning methods for continuous-time Markov decision problems. In *Advances in Neural Information Processing Systems 7: Proc. of the 1994 Conference*, Denver, Colorado, December 1995. MIT Press.

[4] R. A. Brooks. A robust layered control system for a mobile robot. *IEEE Journal of Robotics and Automation*, 2, 1986.

[5] K. W. Currie and A. Tate. O-Plan: the Open Planning Architecture. *Artificial Intelligence*, 52(1), November 1991.

[6] P. Dayan and G. E. Hinton. Feudal reinforcement learning. In Stephen Jose Hanson, Jack D. Cowan, and C. Lee Giles, editors, *Neural Information Processing Systems 5*, San Mateo, California, 1993. Morgan Kaufman.

[7] T. Dean, R. Givan, and S. Leach. Model reduction techniques for computing approximately optimal solutions for markov decision processes. In *Proc. of the Thirteenth Conference on Uncertainty in Artificial Intelligence*, Providence, Rhode Island, August 1997. Morgan Kaufmann.

[8] T. Dean and S.-H. Lin. Decomposition techniques for planning in stochastic domains. In *Proc. of the Fourteenth Int. Joint Conference on Artificial Intelligence*, Montreal, Canada, August 1995. Morgan Kaufmann.

[9] Thomas G. Dietterich. Hierarchical reinforcement learning with the MAXQ value function decomposition. Technical report, Department of Computer Science, Oregon State University, Corvallis, Oregon, 1997.

[10] Y.-J. Hsu. Synthesizing efficient agents from partial programs. In *Methodologies for Intelligent Systems: 6th Int. Symposium, ISMIS '91, Proc.*, Charlotte, North Carolina, October 1991. Springer-Verlag.

[11] T. Jaakkola, M.I. Jordan, and S.P. Singh. On the convergence of stochastic iterative dynamic programming algorithms. *Neural Computation*, 6(6), 1994.

[12] L.-J. Lin. *Reinforcement Learning for Robots Using Neural Networks*. PhD thesis, Computer Science Department, Carnegie-Mellon University, Pittsburgh, Pennsylvania, 1993.

[13] Shieu-Hong Lin. *Exploiting Structure for Planning and Control*. PhD thesis, Computer Science Department, Brown University, Providence, Rhode Island, 1997.

[14] A. McGovern, R. S. Sutton, and A. H. Fagg. Roles of macro-actions in accelerating reinforcement learning. In *1997 Grace Hopper Celebration of Women in Computing*, 1997.

[15] D. Precup and R. S. Sutton. Multi-time models for temporally abstract planning. In *This Volume*.

[16] S. P. Singh. Scaling reinforcement learning algorithms by learning variable temporal resolution models. In *Proceedings of the Ninth International Conference on Machine Learning*, Aberdeen, July 1992. Morgan Kaufmann.

[17] S. P. Singh. Transfer of learning by composing solutions of elemental sequential tasks. *Machine Learning*, 8(3), May 1992.

[18] S. P. Singh, T. Jaakola, and M. I. Jordan. Reinforcement learning with soft state aggregation. In G. Tesauro, D. S. Touretzky, and T. K. Leen, editors, *Neural Information Processing Systems 7*, Cambridge, Massachusetts, 1995. MIT Press.

[19] R. S. Sutton. Temporal abstraction in reinforcement learning. In *Proc. of the Twelfth Int. Conference on Machine Learning*, Tahoe City, CA, July 1995. Morgan Kaufmann.
